# Multi-task Vector Field Learning

[1]Binbin Lin     [2]Sen Yang     [1]Chiyuan Zhang     [2]Jieping Ye     [1]Xiaofei He

[1]State Key Lab of CAD&CG, Zhejiang University, Hangzhou 310058, China
{binbinlinzju, chiyuan.zhang.zju, xiaofeihe}@gmail.com
[2]The Biodesign Institute, Arizona State University, Tempe, AZ, 85287
{senyang, jieping.ye}@asu.edu

## Abstract

Multi-task learning (MTL) aims to improve generalization performance by learning multiple related tasks simultaneously and identifying the shared information among tasks. Most of existing MTL methods focus on learning linear models under the supervised setting. We propose a novel semi-supervised and nonlinear approach for MTL using vector fields. A vector field is a smooth mapping from the manifold to the tangent spaces which can be viewed as a directional derivative of functions on the manifold. We argue that vector fields provide a natural way to exploit the geometric structure of data as well as the shared differential structure of tasks, both of which are crucial for semi-supervised multi-task learning. In this paper, we develop multi-task vector field learning (MTVFL) which learns the predictor functions and the vector fields simultaneously. MTVFL has the following key properties. (1) The vector fields MTVFL learns are close to the gradient fields of the predictor functions. (2) Within each task, the vector field is required to be as parallel as possible which is expected to span a low dimensional subspace. (3) The vector fields from all tasks share a low dimensional subspace. We formalize our idea in a regularization framework and also provide a convex relaxation method to solve the original non-convex problem. The experimental results on synthetic and real data demonstrate the effectiveness of our proposed approach.

## 1 Introduction

In many applications, labeled data are expensive and time consuming to obtain while unlabeled data are abundant. The problem of using unlabeled data to improve the generalization performance is often referred to as semi-supervised learning (SSL). It is well known that in order to make semi-supervised learning work, some assumptions on the dependency between the predictor function and the marginal distribution of data are needed. The *manifold assumption* [15, 5], which has been widely adopted in the last decade, states that the predictor function lives in a low dimensional manifold of the marginal distribution.

Multi-task learning was proposed to enhance the generalization performance by learning multiple related tasks simultaneously. The abundant literature on multi-task learning demonstrates that the learning performance indeed improves when the tasks are related [4, 6, 7]. The key step in MTL is to find the shared information among tasks. Evgeniou *et al.* [12] proposed a regularization MTL framework which assumes all tasks are related and close to each other. Ando and Zhang [2] proposed a structural learning framework, which assumed multiple predictors for different tasks shared a common structure on the underlying predictor space. An alternating structure optimization (ASO) method was proposed for linear predictors which assumed the task parameters shared a low dimensional subspace. Arvind *et al.* [1] generalized the idea of sharing a subspace by assuming that all task parameters lie on a manifold.

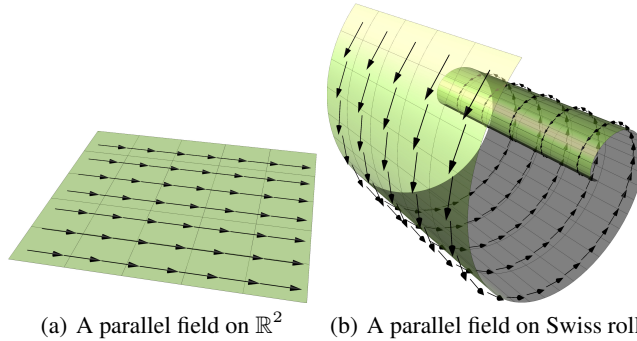

(a) A parallel field on $\mathbb{R}^2$     (b) A parallel field on Swiss roll

Figure 1: Examples of parallel fields. The parallel field on $\mathbb{R}^2$ spans a one dimensional subspace and the parallel field on the Swiss roll spans a two dimensional subspace.

In this paper, we consider semi-supervised multi-task learning (SSMTL). Although many SSL methods have been proposed in the literature [10], these methods are often not directly amenable to MTL extensions [18]. Liu *et al.* [18] proposed an SSMTL framework which encouraged related models to have similar parameters. However they require that related tasks share similar representations [9]. Wang *et al.* [19] proposed another SSMTL method under the assumption that the tasks are clustered [4, 14]. The cluster structure is characterized by task parameters of linear predictor functions. For linear predictors, the task parameters they used are actually the constant gradient of the predictor functions which form a first order differential structure. For general nonlinear predictor functions, we show it is more natural to capture the shared differential structure using vector fields.

In this paper, we propose a novel SSMTL formulation using vector fields. A vector field is a smooth mapping from the manifold to the tangent spaces which can be viewed as a directional derivative of functions on the manifold. In this way, a vector field naturally characterizes the differential structure of functions while also providing a natural way to exploit the geometric structure of data; these are the two most important aspects for SSMTL. Based on this idea, we develop the multi-task vector field learning (MTVFL) method which learns the prediction functions and the vector fields simultaneously. The vector fields we learned are forced to be close to the gradient fields of predictor functions. In each task, the vector field is required to be as parallel as possible. We say that a vector field is parallel if the vectors are parallel along the geodesics on the manifold. In extreme cases, when the manifold is a linear (or an affine) space, then the geodesics of such manifold are straight lines. In such cases, the space spanned by these parallel vectors is a simply one-dimensional subspace. Thus when the manifold is flat (i.e., with zero curvature) or the curvature is small, it is expected that these parallel vectors concentrate on a low dimensional subspace. As an example, we can see from Fig. 1 that the parallel field on the plane spans a one-dimensional subspace and the parallel field on the Swiss roll spans a two-dimensional subspace. For the multi-task case, these vector fields share a low dimensional subspace. In addition, we assume these vector fields share a low dimensional subspace among all tasks. In essence, we use a first-order differential structure to characterize the shared structure of tasks and use a second-order differential structure to characterize the specific parts of tasks. We formalize our idea in a regularization framework and provide a convex relaxation method to solve the original non-convex problem. We have performed experiments using both synthetic and real data; results demonstrate the effectiveness of our proposed approach.

## 2   Multi-task Learning: A Vector Field Approach

In this section, we first introduce vector fields and then present multi-task learning via exploring shared structure using vector fields.

### 2.1   Multi-task Learning Setting and Vector Fields

We first introduce notation and symbols. We are given $m$ tasks, with $n_l$ samples $x_i^l$, $i = 1, \ldots, n_l$ for the $l$-th task. The total number of samples is $n = \sum_l n_l$. For the $l$-th task, we assume the data $\left\{ x_i^l \right\}$ are on a $d_l$-dimensional manifold $\mathcal{M}_l$. All of these data manifolds are embedded in the same

$D$-dimensional ambient space $\mathbb{R}^D$. It is worth noting that the dimensions of different data manifolds are not required to be the same. Without loss of generality, we assume the first $n'_l$ ($n'_l < n_l$) samples are labeled, with $y^l_j \in \mathbb{R}$ for regression and $y^l_j \in \{-1, 1\}$ for classification, $j = 1, \ldots, n'_l$. The total number of labeled samples is $n' = \sum_l n'_l$. For the $l$-th task, we denote the regression function or classification function by $f^*_l$. The goal of semi-supervised multi-task learning is to learn the function value on unlabeled data, i.e., $f^*_l(x^l_i), n'_l + 1 \leq i \leq n_l$.

Given the $l$-th task, we first construct a nearest neighbor graph by either $\epsilon$-neighborhood or $k$ nearest neighbors. Let $x^l_i \sim x^l_j$ denote that $x^l_i$ and $x^l_j$ are neighbors. Let $w^l_{ij}$ denote the weight which measures the similarity between $x^l_i$ and $x^l_j$. It can be approximated by the heat kernel weight or the simple 0-1 weight. For each point $x^l_i$, we estimate its tangent space $T_{x^l_i}\mathcal{M}$ by performing PCA on its neighborhood. We choose the largest $d_l$ eigenvectors as the bases since the tangent space $T_{x^l_i}\mathcal{M}$ has the same dimension as the manifold $\mathcal{M}_l$. Let $T^l_i \in \mathbb{R}^{D \times d_l}$ be the matrix whose columns constitute an orthonormal basis for $T_{x^l_i}\mathcal{M}$. It is easy to show that $P^l_i = T^l_i T^{l^T}_i$ is the *unique* orthogonal projection from $\mathbb{R}^D$ onto the tangent space $T_{x^l_i}\mathcal{M}$ [13]. That is, for any vector $a \in \mathbb{R}^m$, we have $P^l_i a \in T_{x^l_i}\mathcal{M}$ and $(a - P^l_i a) \perp P^l_i a$.

We now formally define the vector field and show how to represent it in the discrete case.

**Definition 2.1** ([16]). *A vector field $X$ on the manifold $\mathcal{M}$ is a continuous map $X : \mathcal{M} \to T\mathcal{M}$ where $T\mathcal{M}$ is the set of tangent spaces, written as $p \mapsto X_p$, with the property that for each $p \in \mathcal{M}$, $X_p$ is an element of $T_p\mathcal{M}$.*

We can think of a vector field on the manifold as an arrow in the same way as we think of the vector field in the Euclidean space, with a given magnitude and direction attached to each point on the manifold, and chosen to be tangent to the manifold. A vector field $V$ on the manifold is called a *gradient field* if there exists a function $f$ on the manifold such that $\nabla f = V$ where $\nabla$ is the covariant derivative on the manifold. Therefore, gradient fields are one kind of vector fields. It plays a critical role in connecting vector fields and functions.

Let $V_l$ be a vector field on the manifold $\mathcal{M}_l$. For each point $x^l_i$, let $V_{x^l_i}$ denote the value of the vector field $V_l$ at $x^l_i$. Recall the definition of vector field, $V_{x^l_i}$ should be a vector in the tangent space $T_{x^l_i}\mathcal{M}_l$. Therefore, we can represent it by the coordinates of the tangent space $T_{x^l_i}\mathcal{M}_l$ as $V_{x^l_i} = T^l_i v^l_i$, where $v^l_i \in \mathbb{R}^{d_l}$ is the local representation of $V_{x^l_i}$ with respect of $T^l_i$. Let $f_l$ be a function on the manifold $\mathcal{M}_l$. By abusing the notation without confusion, we also use $f_l$ to denote the vector $f_l = (f_l(x^1_l), \ldots, f_l(x^l_{n_l}))^T$ and use $V_l$ to denote the vector $V_l = \left( v^l_1{}^T, \ldots, v^l_{n_l}{}^T \right)^T \in \mathbb{R}^{d_l n_l}$. That is, $V_l$ is a $d_l n_l$-dimensional big column vector which concatenates all the $v^l_i$'s for a fixed $l$. Then for each task, we aim to compute the vector $f_l$ and the vector $V_l$.

## 2.2 Multi-task Vector Field Learning

In this section, we introduce multi-task vector field learning (MTVFL).

Many existing MTL methods capture the task relatedness by sharing task parameters. For linear predictors, the task parameters they used are actually the constant gradient vectors of the predictor functions. For general nonlinear predictor functions, we show it is natural to capture the shared differential structure using vector fields. Let $f$ denote the vector $(f^T_1, \ldots, f^T_m)^T$ and $V$ denote the vector $(V^T_1, \ldots, V^T_m)^T = (v^1_1{}^T, \ldots, v^m_{n_l}{}^T)^T$. We propose to learn $f$ and $V$ simultaneously:

- The vector field $V_l$ should be close to the gradient field $\nabla f_l$ of $f_l$, which can be formularized as follows:

$$\min_{f,V} R_1(f, V) = \sum_{l=1}^{m} R_1(f_l, V_l) := \sum_{l=1}^{m} \int_{\mathcal{M}_l} \|\nabla f_l - V_l\|^2. \tag{1}$$

- The vector field $V_l$ should be as parallel as possible:

$$\min_{V} R_2(V) = \sum_{l=1}^{m} R_2(V_l) := \sum_{l=1}^{m} \int_{\mathcal{M}_l} \|\nabla V_l\|^2_{\text{HS}}, \tag{2}$$

where $\nabla$ is the covariant derivative on the manifold, where $\|\cdot\|_{\mathrm{HS}}$ denotes the Hilbert-Schmidt tensor norm [11]. $\nabla V_l$ measures the change of the vector field, therefore minimizing $\int_{\mathcal{M}_l} \|\nabla V_l\|_{\mathrm{HS}}^2$ enforces the vector field $V_l$ to be parallel.

- All vector fields share an $h$-dimensional subspace where $h$ is a predefined parameter:

$$T_i^l v_i^l = u_i^l + \Theta^T w_i^l, \quad s.t.\ \Theta\Theta^T = I_{h\times h}. \tag{3}$$

Since these vector fields are assumed to share a low dimensional space, it is expected that the residual vector $u_i^l$ is small. We define another term $R_3$ to control the complexity as follows:

$$
\begin{aligned}
R_3(v_i^l, w_i^l, \Theta) &= \sum_{l=1}^{m}\sum_{i=1}^{n_l} \alpha\|u_i^l\|^2 + \beta\|T_i^l v_i^l\|^2 \tag{4}\\
&= \sum_{l=1}^{m}\sum_{i=1}^{n_l} \alpha\|T_i^l v_i^l - \Theta^T w_i^l\|^2 + \beta\|T_i^l v_i^l\|^2. \tag{5}
\end{aligned}
$$

Note that $\alpha$ and $\beta$ are pre-specified coefficients, indicating the importance of the corresponding regularization component. Since we would like the vector field to be parallel, the vector norm is not expected to be too small. Besides, we assume the vector fields share a low dimensional subspace, the residual vector $u_i^l$ is expected to be small. In practice we suggest to use a small $\beta$ and a large $\alpha$. By setting $\beta = 0$, $R_3$ will reduce to the regularization term proposed in ASO if we also replace the tangent vectors by the task parameters. Therefore, this formulation is a generalization of ASO.

It can be verified that $w_i^{l^*} = \Theta T_i^l v_i^l = \arg\min_{w_i^l} R_3(v_i^l, w_i^l, \Theta)$. Thus we have $u_i^l = T_i^l v_i^l - \Theta^T w_i^l = (I - \Theta^T\Theta)T_i^l v_i^l$. Therefore, we can rewrite $R_3$ as follows:

$$
\begin{aligned}
R_3(V, \Theta) &= \sum_{l=1}^{m}\sum_{i=1}^{n_l} \alpha\|u_i^l\|^2 + \beta\|T_i^l v_i^l\|^2 \\
&= \sum_{l=1}^{m}\sum_{i=1}^{n_l} \left( \alpha\|(I - \Theta^T\Theta)T_i^l v_i^l\|^2 + \beta\|T_i^l v_i^l\|^2 \right) \tag{6}\\
&= \alpha V^T A_\Theta V + \beta V^T H V,
\end{aligned}
$$

where $H$ is a block diagonal matrix with the diagonal blocks being $T_i^{l^T} T_i^l$, and $A_\Theta$ is another block diagonal matrix with the diagonal blocks being $T_i^{l^T}(I-\Theta^T\Theta)^T(I-\Theta^T\Theta)T_i^l = T_i^{l^T}(I-\Theta^T\Theta)T_i^l$.

Therefore, the proposed formulation solves the following optimization problem:

$$\arg\min_{f, V, \Theta} E(f, V, \Theta) = R_0(f) + \lambda_1 R_1(f, V) + \lambda_2 R_2(V) + \lambda_3 R_3(V, \Theta) \quad s.t.\ \Theta\Theta^T = I_{h\times h}, \tag{7}$$

where $R_0(f)$ is the loss function. For simplicity, we use the quadratic loss function $R_0(f) = \sum_{l=1}^{m}\sum_{i=1}^{n_l'}(f_l(x_i^l) - y_i^l)^2$.

## 2.3 Objective Function in the Matrix Form

To simplify Eq. (7), in this section we rewrite our objective function in the matrix form.

Using the discrete methods in [17], we have the following discrete form equations:

$$
\begin{aligned}
R_1(f_l, V_l) &= \sum_{i\sim j} w_{ij}^l \left( (x_j^l - x_i^l)^T T_i^l v_i^l - f_j^l + f_i^l \right)^2, \tag{8}\\
R_2(f_l, V_l) &= \sum_{i\sim j} w_{ij}^l \left\| P_i^l T_j^l v_j^l - T_i^l v_i^l \right\|^2. \tag{9}
\end{aligned}
$$

Interestingly, with some algebraic transformations, we have the following matrix forms for our objective functions:

$$R_1(f_l, V_l) = 2f_l^T L_l f_l + V_l^T G_l V_l - 2V_l^T C_l f_l, \tag{10}$$

where $L_l$ is the graph Laplacian matrix, $G_l$ is a $d_l n_l \times d_l n_l$ block diagonal matrix, and $C_l = [C_1^{l\,T}, \ldots, C_n^{l\,T}]^T$ is a $d_l n_l \times n_l$ block matrix. Denote the $i$-th $d_l \times d_l$ diagonal block of $G_l$ by $G_{ii}^l$ and the $i$-th $d_l \times n_l$ block of $C_l$ by $C_i^l$, we have

$$G_{ii}^l = \sum_{j \sim i} w_{ij}^l (x_j^l - x_i^l)(x_j^l - x_i^l)^T, \quad C_i^l = \sum_{j \sim i} w_{ij}^l (x_j^l - x_i^l) s_{ij}^l{}^T, \tag{11}$$

where $s_{ij}^l \in \mathbb{R}^{n_l}$ is a selection vector of all zero elements except for the $i$-th element being $-1$ and the $j$-th element being 1. And $R_2$ becomes

$$R_2(V_l) = V_l^T B_l V_l, \tag{12}$$

where $B_l$ is a $d_l n_l \times d_l n_l$ sparse block matrix. If we index each $d_l \times d_l$ block by $B_{ij}^l$, then we have

$$B_{ii}^l = \sum_{j \sim i} w_{ij}^l (Q_{ij}^l Q_{ij}^l{}^T + I), \tag{13}$$

$$B_{ij}^l = \begin{cases} -2w_{ij}^l Q_{ij}^l, & \text{if } x_i \sim x_j \\ 0, & \text{otherwise} \end{cases}, \tag{14}$$

where $Q_{ij}^l = T_i^l{}^T T_j^l$. It is worth nothing that both $R_1$ and $R_2$ depend on tangent spaces $T_i^l$.

Thus we can further write $R_1(f, V)$ and $R_2(V)$ as follows

$$R_1(f, V) = \sum_{l=1}^{m} R_1(f_l, V_l) = 2f^T L f + V^T G V - 2V^T C f, \tag{15}$$

$$R_2(V) = \sum_{l=1}^{m} R_2(V_l) = V^T B V, \tag{16}$$

where $L$, $G$ and $B$ are block diagonal matrices with the corresponding $l$-th block matrix being $L_l$, $G_l$ and $B_l$, respectively. $C$ is a column block matrix with the $l$-th block matrix being $C_l$.

Let $\mathbb{I}$ denote an $n \times n$ diagonal matrix where $\mathbb{I}_{ii} = 1$ if the corresponding $i$-th data is labeled and $\mathbb{I}_{ii} = 0$ otherwise. And let $y \in \mathbb{R}^n$ be a column vector whose $i$-th element is the corresponding label of the $i$-th labeled data and 0 otherwise. Then $R_0(f) = \frac{1}{n'}(f - y)^T \mathbb{I}(f - y)$. Finally, we get the following matrix form for our objective function in Eq. (7) with the constraint $\Theta \Theta^T = I_{h \times h}$ as:

$$E(f, V, \Theta) = R_0(f) + \lambda_1 R_1(f, V) + \lambda_2 R_2(V) + \lambda_3 R_3(V, \Theta)$$

$$= \frac{1}{n'}(f - y)^T \mathbb{I}(f - y) + \lambda_1(2f^T L f + V^T G V - 2V^T C f) + \lambda_2 V^T B V + \lambda_3 V^T(\alpha A_\Theta + \beta H)V$$

$$= \frac{1}{n'}(f - y)^T \mathbb{I}(f - y) + 2\lambda_1 f^T L f + V^T(\lambda_1 G + \lambda_2 B + \lambda_3(\alpha A_\Theta + \beta H))V - 2\lambda_1 V^T C f.$$

It is worth noting that matrices $L, G, B, C$ depend on data, and only the matrix $A_\Theta$ is related to $\Theta$.

## 3 Optimization

In this section, we discuss how to solve the following optimization problem:

$$\arg \min_{f, V, \Theta} E(f, V, \Theta), \quad s.t. \ \Theta \Theta^T = I_{h \times h}. \tag{17}$$

We use the alternating optimization to solve this problem.

- **Optimization of $f$ and $V$.** For a fixed $\Theta$, the optimal $f$ and $V$ can be obtained via solving

$$\arg \min_{f, V} E(f, V, \Theta). \tag{18}$$

- **Optimization of $\Theta$.** For a fixed $V$, the optimal $\Theta$ can be obtained via solving.

$$\arg \min_{\Theta} R_3(V, \Theta), \quad s.t. \ \Theta \Theta^T = I_{h \times h}. \tag{19}$$

## 3.1 Optimization of $f$ and $V$ for a Given $\Theta$

When $\Theta$ is fixed, the objective function is similar to that of the single task case. However, there are some differences we would like to mention. Firstly, when constructing the nearest neighbor graph, data points from different tasks are disconnected. Therefore when estimating tangent spaces, data points from different tasks are independent. Secondly, we do not require the dimension of tangent spaces from each task to be the same.

We note that

$$
\frac{\partial E}{\partial f} = 2\left(\frac{1}{n'}\mathbb{I} + 2\lambda_1 L\right)f - 2\lambda_1 C^T V - 2\frac{1}{n'}y, \tag{20}
$$

$$
\frac{\partial E}{\partial V} = -2\lambda_1 Cf + 2(\lambda_1 G + \lambda_2 H + \lambda_3(\alpha A_\Theta + \beta H))V. \tag{21}
$$

Requiring the derivatives to be vanish, we get the following linear system

$$
\begin{pmatrix} \frac{1}{n'}\mathbb{I} + 2\lambda_1 L & -\lambda_1 C^T \\ -\lambda_1 C & \lambda_1 G + \lambda_2 B + \lambda_3(\alpha A_\Theta + \beta H) \end{pmatrix} \begin{pmatrix} f \\ V \end{pmatrix} = \begin{pmatrix} \frac{1}{n'}y \\ 0 \end{pmatrix}. \tag{22}
$$

Except for the matrix $A_\Theta$, all other matrices can be computed in advance and will not change during the iterative process.

## 3.2 Optimization of $\Theta$ for a Given $V$

Since functions $R_0(f)$, $R_1(f, V)$ and $R_2(V)$ are not related to the variable $\Theta$, we only need to optimize $R_3(V, \Theta)$ subject to $\Theta\Theta^T = I_{h \times h}$.

Recall Eq. (6), we rewrite $R_3(V, \Theta)$ as follows:

$$
\begin{aligned}
\hat{\Theta} &= \arg\min_\Theta \sum_{l=1}^m \sum_{i=1}^{n_l} \alpha\left(\|(I - \Theta^T\Theta)T_i^l v_i^l\|^2 + \frac{\beta}{\alpha}\|T_i^l v_i^l\|^2\right) \\
&= \arg\min_\Theta \alpha\,\mathrm{tr}\left(\mathbf{V}^T((1 + \frac{\beta}{\alpha})I - \Theta^T\Theta)\mathbf{V}\right) \\
&= \arg\max_\Theta \mathrm{tr}(\Theta\mathbf{V}\mathbf{V}^T\Theta^T),
\end{aligned} \tag{23}
$$

where $\mathbf{V} = (T_1^1 v_1^1, \dots, T_{n_m}^m v_{n_m}^m)$ is a $D \times n$ matrix with each column being a tangent vector. The optimal $\hat{\Theta}$ can be obtained by using singular value decomposition (SVD). Let $\mathbf{V} = Z_1 \Sigma Z_2^T$ be the SVD of $\mathbf{V}$ and we assume that the singular values are in a decreasing order in $\Sigma$. Then the rows of $\hat{\Theta}$ are given by the first $h$ columns of $Z_1$.

## 3.3 Convex Relaxation

The orthogonality constraint in Eq. (23) is non-convex. Next, we propose to convert Eq. (23) into a convex formulation by relaxing its feasible domain into a convex set.

Let $\eta = \beta/\alpha$. It can be verified that the following equality holds: $(1 + \eta)I - \Theta^T\Theta = \eta(1 + \eta)(\eta I + \Theta^T\Theta)^{-1}$. Then we can rewrite $R_3(\mathbf{V}, \Theta)$ as $R_3(\mathbf{V}, \Theta) = \alpha\eta(1 + \eta)\,\mathrm{tr}\left(\mathbf{V}^T(\eta I + \Theta^T\Theta)^{-1}\mathbf{V}\right)$. Let $M_e$ be defined as $M_e = \{M : M = \Theta^T\Theta, \Theta\Theta^T = I, \Theta \in \mathbb{R}^{h \times d}\}$. The convex hull [8] of $M_e$ can be expressed as the convex set $M_c$ given by $M_c = \{M : \mathrm{tr}(M) = h, M \preceq I, M \in \mathbb{S}_+^d\}$ and each element in $M_e$ is referred to as an extreme point of $M_c$.

To convert the non-convex problem Eq. (23) into a convex formulation, we replace $\Theta^T\Theta$ with $M$, and naturally relax its feasible domain into a convex set based on the relationship between $M_e$ and $M_c$ presented above; this results in an optimization problem as

$$
\arg\min_\Theta R_3(\mathbf{V}, M), \quad s.t.,\, \mathrm{tr}(M) = h, M \preceq I, M \in \mathbb{S}_+^d, \tag{24}
$$

where $R_3(\mathbf{V}, M)$ is defined as $R_3(\mathbf{V}, M) = \alpha\eta(1 + \eta)\,\mathrm{tr}\left(\mathbf{V}^T(\eta I + M)^{-1}\mathbf{V}\right)$. It follows from [3, Theorem 3.1] that the relaxed $R_3$ is jointly convex in $\mathbf{V}$ and $M$. After we obtain the optimal $M$, the optimal $\Theta$ can be approximated using the first $h$ eigenvectors (corresponding to the largest $h$ eigenvalues) of the optimal $M$.

## 4 Experiments

In this section, we evaluate our method on one synthetic data and one real data set. We compare the proposed Multi-Task Vector Field Learning (MTVFL) algorithm against the following methods: (a) Single Task Vector Field Learning (STVFL, or PFR), (b) Alternating Structure Optimization (ASO) and (c) its nonlinear version - Kernelized Alternating Structure Optimization (KASO). The kernel constructed in KASO uses both labeled data and unlabeled data. Thus it can be viewed as a semi-supervised MTL method.

### 4.1 Synthetic Data

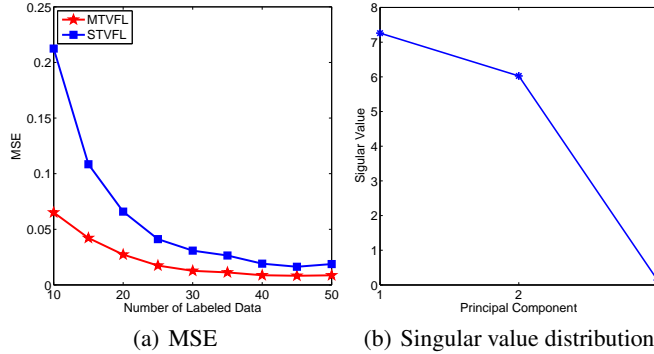

(a) MSE          (b) Singular value distribution

Figure 2: (a) Performance of MTVFL and STVFL; (b) The singular value distribution.

We first construct a synthetic data to evaluate our method in comparison with the semi-supervised single task learning method (STVFL). We generate two data sets including Swiss roll and Swiss roll with hole embedded in 3-dimensional Euclidean space. The Swiss roll is generated by the following equations $x = t_1 \cos t_1; y = t_2; z = t_1 \sin t_1$ where $t_1 \in [3\pi/2, 9\pi/2]; t_2 \in [0, 21]$. The Swill roll with hole excludes points within $t_1 \in [9, 12]$ and $t_2 \in [9, 14]$. The ground truth function is $f(x, y, z) = t_1$. This test is a semi-supervised multi-task regression problem. We randomly select a number of labeled data in each task and try to predict the value on other unlabeled data.

Each data set has 400 points. We construct a nearest neighbor graph for each task. The number of nearest neighbors is set to 5 and the manifold dimension is set to 2 as they are both 2 dimensional manifolds. The shared subspace dimension is set to 2. The regularization parameters are chosen via cross-validation. We perform 100 independent trials with randomly selected labeled sets. The performance is measured by the mean squared error (MSE). We also try ASO and KASO, however they perform poorly since the data is highly nonlinear. The averaged MSE over two tasks is presented in Fig. 2. We can observe that MTVFL consistently outperforms STVFL which demonstrates the effectiveness of SSMTL.

We also show the singular value distribution of the ground truth gradient fields. Given the ground truth $f$, we can compute the gradient field $V$ by taking derivatives of $R_1(f, V)$ with respect to $V$. Requiring the derivative to vanish, we get the following equation $GV = Cf$. After obtaining $V$, the gradient vector $V_{x_i^l}$ at each point can be obtained as $V_{x_i^l} = T_i^l v_i^l$. Then we perform PCA on these vectors and the singular values of the covariance matrix of $V_{x_i^l}$ are shown in Fig. 2 (b). As can be seen from Fig. 2 (b), the number of dominant singular values is 2 which indicates that the ground truth gradient fields concentrate on a 2-dimensional subspace.

### 4.2 Landmine Detection

We use the landmine data set studied in [20]. There are totally 29 sets of data which are collected from various real landmine fields. Each data example is represented by a 9-dimensional vector with a binary label, which is either 1 for landmine or 0 for clutter. The problem of landmine detection

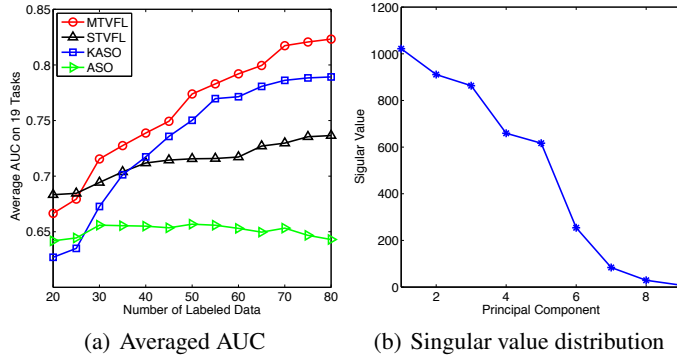

|                    |                              |
|--------------------|------------------------------|
| (a) Averaged AUC   | (b) Singular value distribution |

Figure 3: (a) Performance of various MTL algorithms; (b) The singular value distribution.

is to predict the labels of unlabeled objects. Among the 29 data sets, 1-15 correspond to relatively highly foliated regions and 16-29 correspond to bare earth or desert regions. Following [20], we choose the data sets 1-10 and 16-24 to form 19 tasks.

The basic setup of all the algorithms is as follows. First, we construct a nearest neighbor graph for each task. The number of nearest neighbors is set to 10 and the manifold dimension is set to 4 empirically. These two parameters are the same for all 19 tasks. The shared subspace dimension is set to be 5 for both of MTVFL and ASO and the shared subspace dimension of KASO is set to 10. All the regularization parameters for the four algorithms are chosen via cross-validation. Note that KASO needs to construct a kernel matrix. We use Gaussian kernel in KASO and the Gaussian width is set to be optimal by searching within [0.01, 10].

We perform 100 independent trials with randomly selected labeled sets. We measure the performance by AUC which denotes area under the Receiver Operation Characteristic (ROC) curve. A large AUC value indicates good classification performance. Since the data have severely unbalanced labels, following [20], we do a special setting that assures there is at least one "1" and one "0" labeled sample in the training set of each task. The AUC averaged over the 19 tasks is presented in Fig. 3 (a). As can be seen, MTVFL consistently outperforms the other three algorithms. When the number of labeled data increases, KASO outperforms STVFL. ASO does not improve much when the amount of labeled data increases, which is probably because the data have severely unbalanced labels and the ground truth predictor function is nonlinear. We also show the singular value distribution of the ground truth gradient fields in Fig. 3 (b). The computation of the singular values is the same as in Section. 4.1. As can be seen from Fig. 3 (b), the number of dominant singular values is 5. The percentage of the sum of the first 5 singular values over the total sum is 91.34%, which indicates that the ground truth gradient fields concentrate on a 5-dimensional subspace.

## 5   Conclusion

In this paper, we propose a new semi-supervised multi-task learning formulation using vector fields. We show that vector fields can naturally capture the shared differential structure among tasks as well as the structure of the data manifolds. Our experimental results on synthetic and real data demonstrate the effectiveness of the proposed method. There are several interesting directions suggested in this work. One is the relation between learning on task parameters and learning on vector fields. Ultimately, both of them are learning functions. Another one is to apply other assumptions made in the multi-task learning community into vector field learning, e.g., the cluster assumption.

## Acknowledgments

This work was supported by the National Natural Science Foundation of China under Grants 61125203, 61233011 and 90920303, the National Basic Research Program of China (973 Program) under Grant 2012CB316404, the Fundamental Research Funds for the Central Universities under grant 2011FZA5022, NIH (R01 LM010730) and NSF (IIS-0953662, CCF-1025177).

## Footnotes

[1]The data set is available at `http://www.ee.duke.edu/~lcarin/LandmineData.zip`.

# References

[1] A. Agarwal, H. D. III, and S. Gerber. Learning multiple tasks using manifold regularization. In *Advances in Neural Information Processing Systems 23*, pages 46–54. 2010.

[2] R. K. Ando and T. Zhang. A framework for learning predictive structures from multiple tasks and unlabeled data. *Journal of Machine Learning Research*, 6:1817–1853, 2005.

[3] A. Argyriou, C. A. Micchelli, M. Pontil, and Y. Ying. A spectral regularization framework for multi-task structure learning. In *Advances in Neural Information Processing Systems 20*, pages 25–32. 2008.

[4] B. Bakker and T. Heskes. Task clustering and gating for bayesian multitask learning. *Journal of Machine Learning Research*, 4:83–99, 2003.

[5] M. Belkin, P. Niyogi, and V. Sindhwani. Manifold regularization: A geometric framework for learning from labeled and unlabeled examples. *Journal of Machine Learning Research*, 7:2399–2434, December 2006.

[6] S. Ben-David, J. Gehrke, and R. Schuller. A theoretical framework for learning from a pool of disparate data sources. In *Proceedings of the eighth ACM SIGKDD international conference on Knowledge discovery and data mining*, pages 443–449, 2002.

[7] S. Ben-David and R. Schuller. Exploiting task relatedness for mulitple task learning. In *Conference on Learning Theory*, pages 567–580, 2003.

[8] S. Boyd and L. Vandenberghe. *Convex Optimization*. Cambridge University Press, 2004.

[9] A. Carlson, J. Betteridge, R. C. Wang, E. R. Hruschka, Jr., and T. M. Mitchell. Coupled semi-supervised learning for information extraction. In *Proceedings of the third ACM international conference on Web search and data mining*, pages 101–110, 2010.

[10] O. Chapelle, B. Schölkopf, and A. Zien, editors. *Semi-Supervised Learning*. MIT Press, 2006.

[11] A. Defant and K. Floret. *Tensor Norms and Operator Ideals*. North-Holland Mathematics Studies, North-Holland, Amsterdam, 1993.

[12] T. Evgeniou, C. A. Micchelli, and M. Pontil. Learning multiple tasks with kernel methods. *Journal of Machine Learning Research*, 6:615–637, 2005.

[13] G. H. Golub and C. F. V. Loan. *Matrix computations*. Johns Hopkins University Press, 3rd edition, 1996.

[14] L. Jacob, F. Bach, and J.-P. Vert. Clustered multi-task learning: A convex formulation. In *Advances in Neural Information Processing Systems 21*, pages 745–752. 2009.

[15] J. Lafferty and L. Wasserman. Statistical analysis of semi-supervised regression. In *Advances in Neural Information Processing Systems 20*, pages 801–808, 2007.

[16] J. M. Lee. *Introduction to Smooth Manifolds*. Springer Verlag, New York, 2nd edition, 2003.

[17] B. Lin, C. Zhang, and X. He. Semi-supervised regression via parallel field regularization. In *Advances in Neural Information Processing Systems 24*, pages 433–441. 2011.

[18] Q. Liu, X. Liao, and L. Carin. Semi-supervised multitask learning. In *Advances in Neural Information Processing Systems 20*, pages 937–944. 2008.

[19] F. Wang, X. Wang, and T. Li. Semi-supervised multi-task learning with task regularizations. In *Proceedings of the 2009 Ninth IEEE International Conference on Data Mining*, pages 562–568. IEEE Computer Society, 2009.

[20] Y. Xue, X. Liao, L. Carin, and B. Krishnapuram. Multi-task learning for classification with dirichlet process priors. *Journal of Machine Learning Research*, 8:35–63, 2007.

